# Improving Convergence in Hierarchical Matching Networks for Object Recognition

Joachim Utans*          Gene Gindi[†]
Department of Electrical Engineering
Yale University
P. O. Box 2157 Yale Station
New Haven, CT 06520

## Abstract

We are interested in the use of analog neural networks for recognizing visual objects. Objects are described by the set of parts they are composed of and their structural relationship. Structural models are stored in a database and the recognition problem reduces to matching data to models in a structurally consistent way. The object recognition problem is in general very difficult in that it involves coupled problems of grouping, segmentation and matching. We limit the problem here to the simultaneous labelling of the parts of a single object and the determination of analog parameters. This coupled problem reduces to a weighted match problem in which an optimizing neural network must minimize $E(\mathbf{M}, \mathbf{p}) = \sum_{\alpha i} M_{\alpha i} W_{\alpha i}(\mathbf{p})$, where the $\{M_{\alpha i}\}$ are binary match variables for data parts $i$ to model parts $\alpha$ and $\{W_{\alpha i}(\mathbf{p})\}$ are weights dependent on parameters $\mathbf{p}$. In this work we show that by first solving for estimates $\hat{\mathbf{p}}$ without solving for $M_{\alpha i}$, we may obtain good initial parameter estimates that yield better solutions for $\mathbf{M}$ and $\mathbf{p}$.

[†]Current address: SUNY Stony Brook, Department of Electrical Engineering, Stony Brook, NY 11784

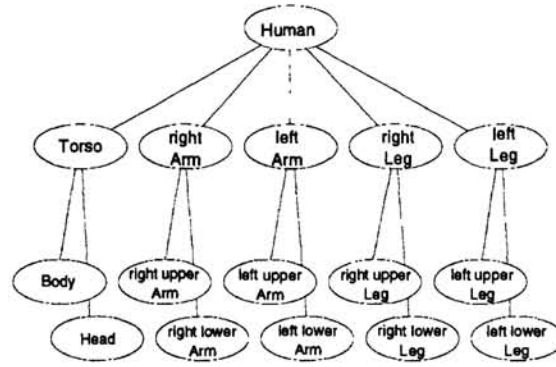

Figure 1: Stored Model for a 3-Level Compositional Hierarchy (compare Figure 3).

# 1    Recognition via Stochastic Forward Models

The Frameville object recognition system introduced by Mjolsness *et al* [5, 6, 1] makes use of a compositional hierarchy to represent stored models. The recognition problem is formulated as the minimization of an objective function. Mjolsness [3, 4] has proposed to derive the objective function describing the recognition problem in a principled way from a stochastic model that describes the objects the system is designed to recognize (*stochastic visual grammar*). The description mirrors the data representation as a compositional hierarchy, at each stage the description of the object becomes more detailed as parts are added.

The stochastic model assigns a probability distribution at each stage of that process. Thus at each level of the hierarchy a more detailed description of parts in terms of their subparts is given by specifying a probability distribution for the coordinates of the subparts. Explicitly specifying these distributions allows for finer control over individual part descriptions than the rather general parameter error terms used before [1, 8]. The goal is to derive a joint probability distribution for an instance of an object and its parts as it appears in the scene. This gives the probability of observing such an object prior to the arrival of the data. Given an observed image, the recognition problem can be stated as a Bayesian inference problem that the neural network solves.

## 1.1    3–Level Stochastic Model

For example, consider the model shown in Figure 1 and 3. The object and its parts are represented as line segments (sticks), the parameters were $\mathbf{p} = (x, y, l, \theta)^T$ with $x, y$ denoting position, $l$ the length of a stick and $\theta$ its orientation. The model considers only a rigid translation of an object in the image.

Only one model is stored. From a central position $\mathbf{p} = (x, y, l, \theta)$, itself chosen from a uniform density, the $N_\beta$ parts at the first level are placed. Their structural relationships is stored as coordinates $\mathbf{u}_\beta$ in an object-centered coordinate frame, i.e. relative to $\mathbf{p}$. While placing the parts, Gaussian distributed noise with mean 0 and is added to the position coordinates to capture the notion of natural variation of the object's shape. The variance is coordinate specific, but we assume the same distribution for the $x$ and $y$ coordinates, $\sigma_{1x}^2$; $\sigma_{1l}^2$ is the variance for the length

component and $\sigma_{1\theta}^2$ for the relative angle. In addition, here we assume for simplicity that all parts are independently distributed. Each of the parts $\beta$ is composed of sub-parts. For simplicity of notation, we assume that each part $\beta$ is composed from the same number of subparts $N_m$ (note that the index $\gamma$ in Figure 2 here corresponds to the double index $\beta m$ to keep track of which part $\beta$ subpart $\beta m$ belongs to on the model side, i.e. the index $\beta m$ denotes the $m^{\text{th}}$ sub-part of part $\beta$). The next step models the *unordering* of parts in the image via a permutation matrix $\mathbf{M}$, chosen with probability $P(\mathbf{M})$, by which their identity is lost. If this step were omitted, the recognition problem would reduce to the problem of estimating part parameters because the parts would already be labeled.

From the grammar we compute the final joint probability distribution (all constant terms are collected in a constant $C$):

$$
\begin{aligned}
P(M, \{\mathbf{p}_{\beta m}\}, \{\mathbf{p}_\beta\}, \mathbf{p}) = \\
C \exp \Bigg( \sum_\beta \Bigg( &-\frac{1}{2\sigma_{\beta x}^2}(x_\beta - (x + u_{\beta x}))^2 - \frac{1}{2\sigma_{\beta x}^2}(y_\beta - (y + u_{\beta y}))^2 \\
&-\frac{1}{2\sigma_{\beta l}^2}(l_\beta - (l + u_{\beta l}))^2 - \frac{1}{2\sigma_{\beta \theta}^2}(\theta_\beta - (\theta + u_{\beta\theta}))^2 \Bigg) \Bigg) \\
\exp \Bigg( \sum_{\beta m\, k} M_{\beta m k} \Bigg( &-\frac{1}{2\sigma_{\beta m x}^2}(x_k - (x_\beta + u_{\beta m x}))^2 - \frac{1}{2\sigma_{\beta m x}^2}(y_k - (y_\beta + u_{\beta m y}))^2 \\
&-\frac{1}{2\sigma_{\beta m l}^2}(l_k - (l_\beta + u_{\beta m l}))^2 - \frac{1}{2\sigma_{\beta m \theta}^2}(\theta_k - (\theta_\beta + u_{\beta m \theta}))^2 \Bigg) \Bigg)
\end{aligned}
\tag{1}
$$

## 1.2 Frameville Architecture for Part Labelling within a single Object

The stochastic forward model for the part labelling problem with only a single object present in the scene translates into a reduced Frameville architecture as depicted in Figure 2. The compositional hierarchy parallels the steps in the stochastic model as parts are added at each level. Match variables appear only at the lowest level, corresponding to the permutation step of the grammar. Parts in the image must be matched to model parts and parts found to belong to the stored object must be grouped together.

The single match neuron $M_{\alpha i}$ at the highest level can be set to unity since we assume we know the object's identity and only a single object is present. Similarly, all terms $ina_{ij}$ from the first to the second level can be set to unity for the correct grouping since the grouping is known at this point from the forward model description. In addition, at the intermediate (second) level, we may set all $M_{\beta j} = 1$ for $\beta = j$ and $M_{\beta j} = 0$ otherwise with no loss of generality. These mid–level frames may be matched ahead of time, but their parameters must be computed from data. Introducing a part permutation at the intermediate levels thus is redundant. Given this, an additional simplification **ina** grouping variables at the lowest (third) level is possible. Since parts are pre-matched at all but the lowest level, $ina_{jk}$ can be expressed in terms of the part match $M_{\gamma k}$ as $ina_{jk} = M_{\gamma k} INA_{\gamma \beta} M_{\beta j}$ and explicitly representing $ina_{jk}$ as variables is not necessary.

The input to the system are the $\{\mathbf{p}_k\}$, recognition involves finding the parameters

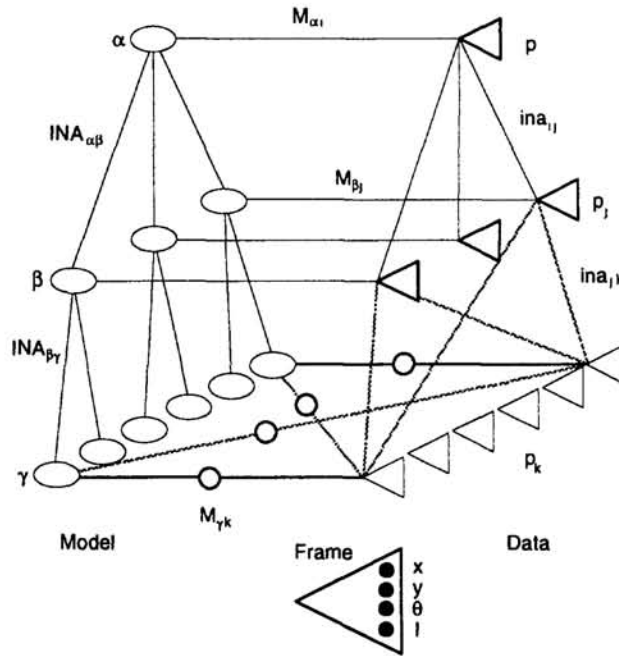

Figure 2: Frameville Architecture for the Stochastic Model. The 3–level grammar leads to a reduced "Frameville" style network architecture: a single model is stored on the model side and only one instance of the model is present in the input data. The ovals on the model side represent the object, its parts and subparts (compare Figure 1); the arcs *INA* represent their structural relationship. On the data side, the triangles represent parameter vectors (or *frames*) describing an instance of the object in the scene. At the lowest level the $p_k$ represent the input data, parameters at higher levels in the hierarchy must be computed by the network (represented as bold triangles). *ina* represents the grouping of parts on the data side (see text). The horizontal lines represent assignments from frames on the data side to nodes on the model side. At the intermediate level, frames are prematched to the corresponding parts on the model side; match variables are necessary only at the lowest level (represented as bold lines with circles).

$\mathbf{p}$ and $\{\mathbf{p}_j\}$ as well as the labelling of parts $\mathbf{M}$. Thus, from Bayes Theorem

$$
\begin{aligned}
P(\mathbf{M}, \mathbf{p}, \{\mathbf{p}_j\}|\{\mathbf{p}_k\}) &= \frac{P(\{\mathbf{p}_k\}|\mathbf{M}, \mathbf{p}, \{\mathbf{p}_j\})P(\mathbf{M}, \mathbf{p}, \{\mathbf{p}_j\})}{P(\{\mathbf{p}_k\})} \\
&\propto P(\mathbf{M}, \mathbf{p}, \{\mathbf{p}_j\}, \{\mathbf{p}_k\})
\end{aligned}
\tag{2}
$$

and recognition reduces to finding the most probable values for $\mathbf{p}$, $\{\mathbf{p}_j\}$ and $\mathbf{M}$ given the data:

$$
\underset{\mathbf{M}, \mathbf{p}, \{\mathbf{p}_j\}}{\arg \max} \quad P(\mathbf{M}, \mathbf{p}, \{\mathbf{p}_j\}, \{\mathbf{p}_k\})
\tag{3}
$$

Solving the inference problem involves finding the MAP estimate and is is equivalent to minimizing the exponent in equation (1) with respect to $\mathbf{M}$, $\mathbf{p}$ and $\{\mathbf{p}_j\}$.

## 2    Bootstrap: Coarse Scale Hints to Initialize the Network

### 2.1    Compositional Hierarchy and Scale Space

In some labelling approaches found in the vision literature, an object is first labelled at the coarse, low resolution, level and approximate parameters are found. In this top–down approach the information at the higher, more abstract, levels is used

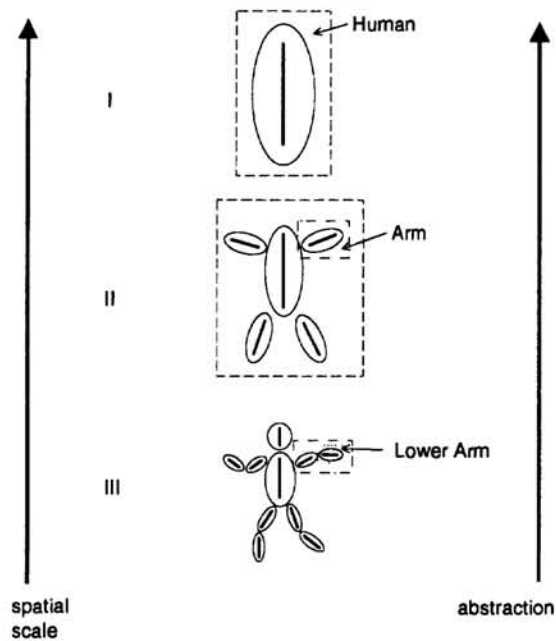

Figure 3: Compositional Hierarchy vs. Scale Space Hierarchy. A compositional hierarchy can represent a scale space hierarchy. At successive levels in the hierarchy, more and more detail is added to the object.

to select initial values for the parts at the next lower level of abstraction. The segmentation and labelling at this next lowest level is thus not done blindly; rather it is strongly influenced contextually by the results at the level above.

In fact, in very general terms such a scheme was described by Marr and Nishihara [2]. They advocate in essence a hierarchical model base in which a shape is first matched to the highest levels, and defaults in terms of relative object–based parameters of parts at the next level are recalled from memory. These defaults then serve as initial values in an unspecified segmentation algorithm that derives part parameters; this step is repeated recursively until the lowest level is reached.

Note that the highest level of abstractions correspond to the coarsest levels of spatial scale. There is nothing in the design of the model base that demands this, but invariably, elements at the top of a compositional hierarchy are of coarser scale since they must both include the many subparts below, and summarize this inclusion with relatively few parameters. Figure 3 illustrates the correspondence between these representations. In this sense, the compositional hierarchy as applied to shapes includes a notion of scale, but there is no "scale–space" operation of intentionally blurring data. The notion of Scale Space as utilized here thus differs from the application of the method to low-level computations in the visual domain where auxiliary coarse scale representations are computed explicitly. The object representations in the Frameville system as described earlier combines both, bottom–up and top–down elements. If the top–down aspects of the scheme described by Marr and Nishihara [2] could be incorporated into the Frameville architecture, then our previous simulation results [8] suggest that much better performance can be expected from the neural network. Two problems must be addressed: (1) How do we obtain, from the observed raw data alone, a coarse estimate of the slot parameters at the highest level and (2) given these crude estimates how do we utilize them to recall default settings for the segmentation one level below?

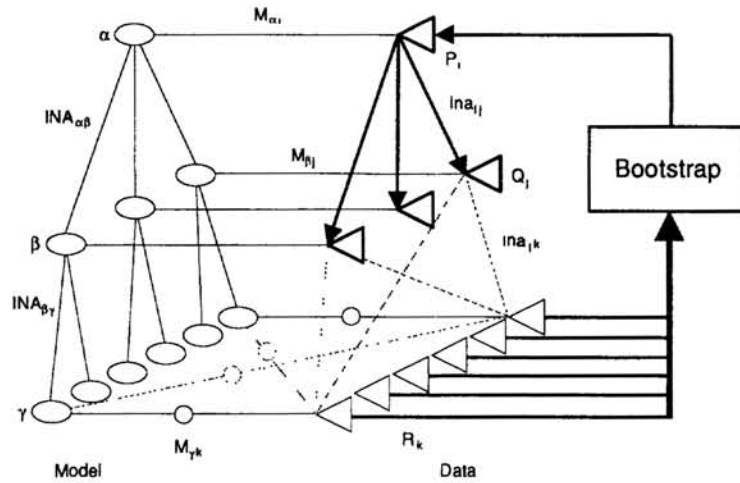

Figure 4: Bootstrap computation for a network from a 3–level grammar. Analog frame variables at the top and intermediate level are initialized from data by a bootstrap computation (bold lines indicate the flow of information)

## 2.2   Initialization of Coarse Scale Parameters

We propose to aid convergence by supplying initial values for the analog variables $\mathbf{p}$ and $\{\mathbf{p}_j\}$; these must be computed from data without making use of the labelling. In general, it is not possible to solve for the analog parameters without knowledge of the correct permutation matrix $\mathbf{M}$. However, for the purpose of obtaining an approximation $\hat{\mathbf{p}}$ one can derive a new objective function that does not depend on $\mathbf{M}$ and the parameters $\{\mathbf{p}_j\}$ by integrating over the $\{\mathbf{p}_j\}$ and summing over all possible permutation matrices $\mathbf{M}$:

$$P(\mathbf{p}, \{\mathbf{p}_k\}) = \sum_{\substack{\{\mathbf{M}\}\,|\,\mathbf{M}\ \text{is a}\\ \text{permutation}}} \int d\{\mathbf{p}_j\} P(\mathbf{p}, \{\mathbf{p}_j\}, \{\mathbf{p}_k\}, \mathbf{M}) \tag{4}$$

This formulation leads to an Elastic Net type network [9, 7]. However, this implementation of a separate network for the bootstrap computations is expensive.

Here we use simpler computation where the coarse scale parameters are estimated by computing sample averages, corresponding to finding the solution for the Elastic Net in the high temperature limit [7]. For the position $\mathbf{x}$ we find, after integrating over the $\{\mathbf{x}_j\}$,

$$x = \frac{1}{\sum_{\beta m} 1/(\sigma_{\beta x}^2 \sigma_{\beta m x}^2)} \sum_{\beta m\,k} \frac{M_{\beta m k}\, x_k}{\sigma_{\beta x}^2 \sigma_{\beta m x}^2} - \frac{1}{\sum_{\beta m} 1/(\sigma_{\beta x}^2 \sigma_{\beta m x}^2)} \sum_{\beta m} \frac{u_{\beta m x}}{\sigma_{\beta x}^2 \sigma_{\beta m x}^2}$$

$$- \frac{1}{\sum_{\beta} 1/\sigma_{\beta x}^2} \sum_{\beta} \frac{u_{\beta x}}{\sigma_{\beta x}^2} \tag{5}$$

and similarly for $y$. Since the assignment $M_{\beta m\,k}$ of subparts $k$ on the data side to subparts $\beta m$ on the model side is not known at this point, the first term in equations (5) cannot be evaluated. After approximating the actual variance with

an average variance, these equations reduce to

$$\hat{x} = \frac{1}{N_\beta N_m} \sum_k x_k - \frac{1}{N_\beta N_m} \sum_{\beta m} u_{\beta m x} - \frac{1}{N_\beta} \sum_\beta u_{\beta x} \qquad (6)$$

In terms of the objective function this translates into assuming that here the error terms for all parts are weighted equally. Since these weights would depend on the actual part match, this just corresponds to our ignorance regarding identity of the parts. This approximation assumes that the variances do not differ by a large amount, otherwise the approximation $\hat{p}$ will not be close to the true values. Since the model can be designed such that the part primitives used at the lowest level of the grammar are not highly specialized as would be the case for abstractions at higher levels of the model, the approximation proved sufficient for the problems studied here.

The neural network can be used to perform the calculation. The Elastic Net formulation assigns approximately equal weights to all possible assignments at high temperatures. Thus, this behavior can be expressed in the original network with match variables by choosing $M_{\beta m k} = 1/(N_\beta N_m) \; \forall \; i,j$. This leads to the following two-pass bootstrap computation. Using this specific choice for $M$ only the analog variables need to be updated to compute the coarse scale estimates. The network with constant $M$ is just the neural network implementation for computing $\hat{x}$ from equation (6). After these have converged, $\hat{x}$ can be used to compute $\hat{x}_j = \hat{x} + u_\beta$. Thus, the parameters for intermediate levels can by hypothesized from the coarse scale estimate $\hat{x}$ by adding the known transformation (recall that for intermediate levels, the part identity is preserved and no permutation steps takes place (see Figure 2)). Then the network is restarted with random values for the match variables to compute the correct labelling and the correct parameters.

## 2.3   Simulation Results

The bootstrap procedure has been implemented for a 3-level hierarchical model. The model describes a "gingerbread man" as shown in Figure 3. The incorrect solutions observed did not, in the vast majority of cases, violate the permutation matrix constraint, i.e. the assignment was unique. However, even though the assignment is unique, parts where not always assigned correctly. Most commonly, the identity of neighboring parts was interchanged, in particular for cases with large variance.

The advantage of using the bootstrap initialization is clear from Figure 5. For the simulation, $\sigma_2^2 = 2\sigma_1^2$; the noise variance was identical for all parts. The network computed the solution reliably for large noise variances. In such cases the performance of the network without initialization deteriorates rapidly. Only one set of 10 experiments was used for the graph but in all simulations performed, the network with initialization consistently outperformed the network without initialization. Figure 5(right) shows the time measured in the number of iterations necessary for the network to converge; it is almost unaffected by the increase in the noise variance. This is because the initial values derived from data are still close to the final solution. While in some cases, the random starting point happens to be close to the correct solution and the network without initialization converges rapidly, Figure 5 reflect the typical behavior and demonstrate the advantage of computing approximate initial values.

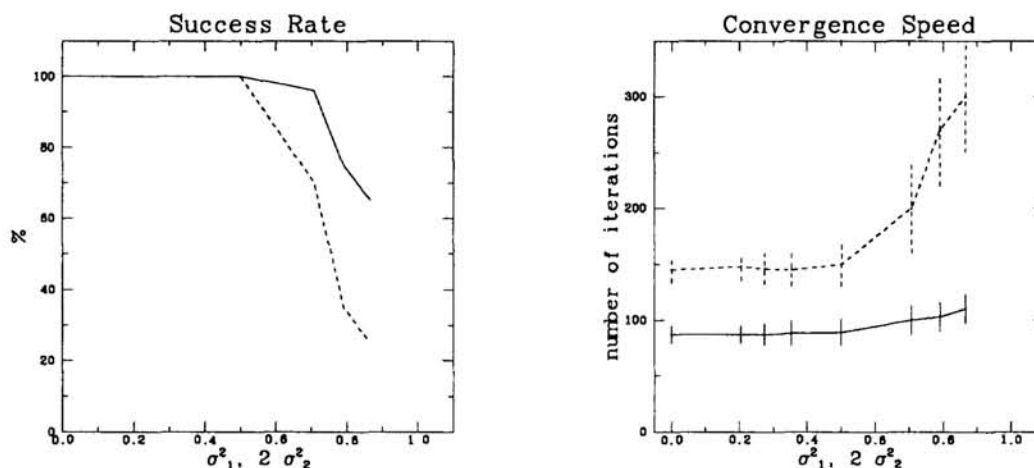

Figure 5: Results Comparing the Network without and with Initialization (solid line).
Left: The success rate indicates the rate at which the network converged to the correct solutions. $\sigma_1^2$ denotes the noise variance at the intermediate level of the model and $\sigma_2^2$ the noise variance at the lowest level. Only one set of 10 experiments was used for the graph but in all simulations performed, the network with initialization consistently outperformed the network without initialization.
Right: The graph shows the average time it takes for the network to converge (as measured by the number of iterations) averaged over 10 experiments. Only simulations where the network converged to the correct solution are used to compute the average time for convergence. The stopping criterion used required all the match neurons to assume values $M_{1j} > 0.95$ or $M_{1j} < 0.05$. The error bars denote the standard deviation.

## Acknowledgements

This work was supported in part by AFOSR grant AFOSR 90-0224. We thank E. Mjolsness and A. Rangarajan for many helpful discussions.

## Footnotes

*Current address: International Computer Science Institute, 1947 Center Street, Suite 600, Berkeley, CA 94704, utans@icsi.berkeley.edu

## References

[1] G. Gindi, E. Mjolsness, and P. Anandan. Neural networks for model based recognition. In *Neural Networks: Concepts, Applications and Implementations*, pages 144–173. Prentice–Hall, 1991.

[2] David Marr. *Vision*. W. H. Freeman and Co., New York, 1982.

[3] E. Mjolsness. Bayesian inference on visual grammars by neural nets that optimize. Technical Report YALEU–DCS–TR–854, Yale University, Dept. of Computer Science, 1991.

[4] E. Mjolsness. Visual grammars and their neural nets. In R.P. Lippmann J.E. Moody, S.J. Hanson, editor, *Advances in Neural Information Processing Systems 4*. Morgan Kaufmann Publishers, San Mateo, CA, 1992.

[5] Eric Mjolsness, Gene Gindi, and P. Anandan. Optimization in model matching and perceptual organization: A first look. Research report yaleu/dcs/rr-634, Yale University, Department of Computer Science, 1988.

[6] Eric Mjolsness, Gene R. Gindi, and P. Anandan. Optimization in model matching and perceptual organization. *Neural Computation*, vol. 1, no. 2, 1989.

[7] Joachim Utans. *Neural Networks for Object Recognition within Compositional Hierarchies*. PhD thesis, Department of Electrical Engineering, Yale University, New Haven, CT 06520, 1992.

[8] Joachim Utans, Gene R. Gindi, Eric Mjolsness, and P. Anandan. Neural networks for object recognition within compositional hierarchies: Initial experiments. Technical report 8903, Yale University, Center for Systems Science, Department Electrical Engineering, 1989.

[9] A. L. Yuille. Generalized deformable models, statistical physics, and matching problems. *Neural Computation*, 2(2):1–24, 1990.